# Observability of Neural Network Behavior

**Max Garzon**
garzonm@hermes.msci.memst.edu
Institute for Intelligent Systems

**Fernanda Botelho**
botelhof@hermes.msci.memst.edu
Department of Mathematical Sciences

Memphis State University
Memphis, TN 38152  U.S.A.

## Abstract

We prove that except possibly for small exceptional sets, discrete-time analog neural nets are globally observable, i.e. all their corrupted pseudo-orbits on computer simulations actually reflect the true dynamical behavior of the network. Locally finite discrete (boolean) neural networks are observable without exception.

## 1  INTRODUCTION

We address some aspects of the general problem of implementation and robustness of (mainly recurrent) autonomous discrete-time neural networks with continuous activation (herein referred to as analog networks) and discrete activation (herein, boolean networks). There are three main sources of perturbations from ideal operation in a neural network. First, the network's parameters may have been contaminated with noise from external sources. Second, the network is being implemented in optics or electronics (digital or analog) and inherent measurement limitations preclude the use of perfect information on the network's parameters. Third, as has been the most common practice so far, neural networks are simulated or implemented on a digital device, with the consequent limitations on precision to which net parameters can be represented. Finally, for these or other reasons, the activation functions (e.g. sigmoids) of the network are not known precisely or cannot be evaluated properly. Although perhaps negligible in a single iteration, these perturbations are likely to accumulate under iteration, even in feedforward nets. Eventually, they may, in fact, distort the results of the implementation to the point of making the simulation useless, if not misleading.

There is, therefore, an important difference between the intended operation of an idealized neural network and its observable behavior. This is a classical problem in systems theory and it has been addressed in several ways. First, the classical notions of *distinguishability and observability* in control theory (Sontag, 1990) which roughly state that every pair of system's *states* are distinguishable by different outputs over evolution in finite time. This is thus a notion of *local* state observability. More recently, several results have established more global notions of *identifiability* of discrete-time feedfoward (Sussmann, 1992; Chen, Lu, Hecht-Nelson, 1993) and continuous-time recurrent neural nets (Albertini and Sontag, 1993a,b), which roughly state that for given odd activation functions (such as *tanh*), the weights of a neural network are essentially uniquely determined (up to permutation and cell redundancies) by the input/output behavior of the network. These notions do assume error-free inputs, weights, and activation functions.

In general, a computer simulation of an orbit of a given dynamical system in the continuuum (known as a pseudo-orbit) is, in fact, far from the orbit in the ideal system. Motivated by this problem, Birkhoff introduced the so-called *shadowing property*. A system satisfies the *shadowing property* if *all* pseudo-orbits are uniformly approximated by actual orbits so that the former capture the long-term behavior of the system. Bowen showed that sufficiently hyperbolic systems in real euclidean spaces do have the shadowing property (Bowen, 1978). However, it appears difficult even to give a characterization of exactly which maps on the interval possess the property –see e.g. (Coven, Kan, Yorke, 1988). Precise definitions of all terms can be found in section 2.

By comparison to state observability and identifiability, the shadowing property is a type of *global observability* of a system through its dynamical behavior. Since autonomous recurrent networks can be seen as dynamical systems, it is natural to investigate this property. Thus, a neural net is *observable* in the sense that its behavior (i.e. the sequence of its ideal actions on given initial conditions) can be observed on computer simulations or discrete implementations, despite inevitable concomitant approximations and errors.

The purpose of this paper is to explore this property as a deterministic model for perturbations of neural network behavior in the presence of arbitrary small errors from various sources. The model includes both discrete and analog networks. In section 4 we sketch a proof that locally finite boolean neural networks (even with an infinite number of neurons), are *all* observable in this sense. This is not true in general for analog networks, and section 3 is devoted to sketching necessary and sufficient conditions for the relatively few analog exceptions for the most common transfer functions: hard-thresholds, a variety of sigmoids (hyperbolic tangent, logistic, etc.) and saturated linear maps. Finally, section 5 discusses the results and poses some other problems worthy of further research.

## 2   DEFINITIONS AND MAIN RESULTS

This section contains notation and precise definitions in a general setting, so as to include discrete-time networks both with discrete and continuous activations.

Let $f : X \to X$ be a continuous map of a compact metric space with metric $| *, * |$.

The *orbit* of $x \in X$ is the sequence $\{x, f(x), \ldots, f^k(x) \ldots\}$, i.e. a sequence of points $\{x^k\}_{k \geq 0}$ for which $x^{k+1} = f(x^k)$, for all $k \geq 0$. Given a number $\delta > 0$, a $\delta$-*pseudo-orbit* is a sequence $\{x^k\}$ so that the distances $|f(x^k), x^{k+1}| < \delta$ for all $k \geq 0$. Pseudo-orbits arise as trajectories of ideal dynamical processes contaminated by errors and noise. In such cases, especially when errors propagate exponentially, it is important to know when the numerical process is actually representing some meaningful trajectory of the real process.

**Definition 2.1** *The map $f$ on a metric space $X$ is (globally) observable (equivalently, has the shadowing property, or is traceable) if and only if for every $\epsilon > 0$ there exists a $\delta > 0$ so that any $\delta$-pseudo-orbit $\{x^k\}$ is $\epsilon$-approximated by the orbit, under $f$, of some point $z \in X$, i.e. $|x^k, f^k(z)| < \epsilon$ for all $k \geq 0$.*

One might observe that computer simulations only run for finite time. On compact spaces (as is the case below), observability can be shown to be equivalent to a similar property of shadowing finite pseudo-orbits.

'Analog neural network' here means a finite number $n$ of units (or cells), each of which is characterized by an *activation* (sometimes called output) function $\sigma_i :$ $\mathbf{R} \to \mathbf{R}$, and *weight* matrix $W$ of synaptic strengths between the various units. Units can assume real-valued activations $x_i$, which are updated synchronously and simultaneously at discrete instants of time, according to the equation

$$x_i(t+1) \quad = \quad \sigma_i[\sum_j w_{ij} x_j(t)]. \tag{1}$$

The total activation of the network at any time is hence given by a vector $x$ in euclidean space $\mathbf{R}^n$, and the entire network is characterized by a global dynamics

$$T(x) \quad = \quad \sigma[Wx], \tag{2}$$

where $Wx$ denotes ordinary product and $\sigma$ is the map acting as $\sigma_i$ along the $i$th component. This component in a vector $x$ is denoted $x_i$ (as opposed to $x^k$, the $k^{th}$ term of a sequence). The unit hypercube in $\mathbf{R}^n$ is denoted $I^n$. An analog network is then defined as a dynamical system in a finite-dimensional euclidean space and one may then call a neural network (globally) observable if its global dynamics is an observable dynamical system. Likewise for boolean networks, which will be defined precisely in section 4.

We end this section with some background facts about observability on the continuum. It is perhaps surprising but a trivial remark that the identity map of the real interval is not observable in this sense, since orbits remain fixed, but pseudo-orbits may drift away from the original state and can, in fact, be dense in the interval. Likewise, common activation functions of neural networks (such as hard thresholds and logistic maps) are not observable. For linear maps, observability has long been known to be equivalent to *hyperbolicity* (all eigenvalues $\lambda$ have $|\lambda| \neq 1$). Composition of observable maps is usually not observable (take, for instance, a hyperbolic homeomorphism and its inverse). In contrast, composition of linear and nonobservable activation functions in neural networks are, nevertheless, observable. The main take-home message can be loosely summarized as follows.

**Theorem 2.1** *Except for a negligible fraction of exceptions, discrete-time analog neural nets are observable. All discrete (boolean) neural networks are observable.*

# 3  ANALOG NEURAL NETWORKS

This section contains (a sketch) of necessary and sufficient conditions for analog networks to be observable for common types of activations functions.

## 3.1  HARD-THRESHOLD ACTIVATION FUNCTIONS

It is not hard to give necessary and sufficient conditions for observability of nets with discrete activation functions of the type

$$\sigma_i(u) \quad := \quad \left\{ \begin{array}{ll} 1 & \text{if } u \geq \theta_i \\ 0 & \text{else.} \end{array} \right.$$

where $\theta_i$ is a theshold characterizing cell $i$.

**Lemma 3.1** *A map* $f : \mathbf{R}^n \to \mathbf{R}^n$ *with finite range is observable if and only if it is continuous at each point of its range.*

PROOF. The condition is clearly sufficient. If $f$ is continuous at every point of its range, small enough perturbations $x^{k+1}$ of an image $f(x^k)$ have the same image $f(x^{k+1}) = f(f(x^k))$ and hence, for $\delta$ small enough, every $\delta$—pseudo-orbit is traced by the first element of the pseudo-orbit. Conversely, assume $f$ is not continuous at a point of its range $f(x^0)$. Let $x^1, x^2, \ldots$ be a sequence constant under $f$ whose image does not converge to $f(f(x^0))$ (such a sequence can always be so chosen because the range is discrete). Let

$$\epsilon := \frac{1}{2} \min_{x,y \in \mathbf{R}^n} |f(x), f(y)|.$$

For a given $\delta > 0$ the pseudo-orbit $x^0, x^k, f(x^k), f^2(x^k), \ldots$ is not traceable for $k$ large enough. Indeed, for any $z$ within $\epsilon$-distance of $x^0$, either $f(z) \neq f(x^0)$, in which case this distance is at least $\epsilon$, or else they coincide, in which case $|f^2(z), f(x^k)| > \epsilon$ anyway by the choice of $x^k$. □

Now we can apply Lemma 3.1 to obtain the following characterization.

**Theorem 3.1** *A discrete-time neural net $T$ with weight matrix $W := (w_{ij})$ and threshold vector $\theta$ is observable if and only if for every $y$ in the range of $T$, $(Wy)_i \neq \theta_i$ for every $i$ $(1 \leq i \leq n)$.*

## 3.2  SIGMOIDAL ACTIVATION FUNCTIONS

In this section, we establish the observability of arbitrary neural nets with a fairly general type of sigmoidal activation functions, as defined next.

**Definition 3.1** *A map $\sigma : \mathbf{R} \to \mathbf{R}$ is sigmoidal if it is strictly increasing, bounded (above and below), and continuously differentiable.*

Important examples are the logistic map

$$\sigma_i(u) = \frac{1}{1 + \exp(-\mu u)},$$

the arctan and the hyperbolic tangent maps

$$\sigma_i(u) = arctan(\mu u) \qquad , \quad \sigma_i(u) = tanh(u) = \frac{\exp(u) - \exp(-u)}{\exp(u) + \exp(-u)} .$$

Note that, in particular, the range of a sigmoidal map is an open and bounded interval, which without loss of generality, can be assumed to be the unit interval $I$. Indeed, if a neural net has weight matrix $W$ and activation function $\sigma$ which is conjugate to an activation function $\sigma'$ by a conjugacy $\phi$, then

$$\sigma \circ W \sim \sigma' \phi W \phi^{-1}$$

where $\sim$ denotes conjugacy. One can, moreover, assume that the gain factors in the sigmoid functions are all $\mu = 1$ (multiply the rows of $W$).

**Theorem 3.2** *Every neural networks with a sigmoidal activation function has a strong attractor, and in particular, it is observable.*

PROOF. Let a neural net with $n$ cells have weight matrix $W$ and sigmoidal $\sigma$. Consider a parametrized family $\{T_\mu\}_\mu$ of nets with sigmoidals given by $\sigma_\mu := \mu\sigma$. It is easy to see that each $T_\mu$ $(\mu > 0)$ is conjugate to $T$. However, $W$ needs to be replaced by a suitable conjugation with a homeomorphism $\phi_\mu$. By Brouwer's fixed point theorem, $T_\mu$ has a fixed point $p^*$ in $I^n$. The key idea in the proof is the fact that the dynamics of the network admits a Lyapunov function given by the distance from $p^*$. Indeed,

$$\parallel T_\mu(x) - T_\mu(p^*) \parallel \le \sup_y |JT_\mu| \parallel x - p^* \parallel,$$

where $J$ denotes jacobian. Using the chain rule and the fact that the derivatives of $\phi_\mu$ and $\sigma_\mu$ are bounded (say, below by $b$ and above by $B$), the Jacobian satisfies

$$|JT_\mu(y)| \le \mu^n (bB)^n |W|,$$

where $|W|$ denotes the determinant of $W$. Therefore we can choose $\mu$ small enough that the right-hand side of this expression is less than 1 for arbitrary $y$, so that $T_\mu$ is a contraction. Thus, the orbit of the first element in any $\epsilon$-pseudo-orbit $\epsilon$-traces the orbit.  $\square$

## 3.3  SATURATED-LINEAR ACTIVATION FUNCTIONS

The case of the nondifferentiable saturated linear sigmoid given by the piecewise linear map

$$\sigma_i(u) \quad := \quad \begin{cases} 0, & \text{for } u < 0 \\ u, & \text{for } 0 \le u \le 1 \\ 1, & \text{for } u > 1 \end{cases} \tag{3}$$

presents some difficulties. First, we establish a general necessary condition for observability, which follows easily for linear maps since shadowing is then equivalent to hyperbolicity.

**Theorem 3.3** *If $T$ leaves a segment of positive length pointwise fixed, then $T$ is not observable.*

Although easy to see in the case of one-dimensional systems due to the fact that the identity map is not observable, a proof in higher dimensions requires showing that a dense pseudo-orbit in the fixed segment is not traceable by points outside the segment. The proof makes use of an auxiliary result.

**Lemma 3.2** *A linear map $L : \mathbf{R}^n \to \mathbf{R}^n$, acts along the orbit of a point $x$ in the unit hypercube either as an attractor to 0, a repellor to infinity, or else as a rigid rotation or reflection.*

PROOF. By passing to the complexification $L' : \mathbf{C}^n \to \mathbf{C}^n$ of $L$ and then to a conjugate, assume without loss of generality that $L$ has a matrix in Jordan canonical form with blocks either diagonal or diagonal with the first upper minor diagonal of 1s. It suffices to show the claim for each block, since the map is a cartesian product of the restrictions to the subspaces corresponding to the blocks. First, consider the diagonal case. If the eigenvalues $|\lambda| < 1$ ($|\lambda| > 1$, respectively), clearly the orbit $L^k(x) \to 0$ ($\| L^k(x) \| \to \infty$). If $|\lambda| = 1$, $L$ acts as a rotation. In the nondiagonal case, it is easy to see that the iterates of $x = (x_1, \cdots, x_m)$ are given by

$$\mathbf{L}^t(x) \quad := \quad \sum_{k=0}^{t} \binom{t}{k} \lambda^{t-k} x_{k+1} + \sum_{k=0}^{t-1} \binom{t}{k} \lambda^{t-k} x_{k+2} + \cdots + \lambda^t x_m . \qquad (4)$$

The previous argument for the diagonal block still applies for $|\lambda| \neq 1$. If $|\lambda| = 1$ and if at least two components of $x \in I^n$ are nonzero, then they are positive and again $\| L(x) \| \to \infty$. In the remaining case, $L$ acts as a rotation since it reduces to multiplication of a single coordinate of $x$ by $\lambda$. $\square$

PROOF OF THEOREM 3.3. Assume that $T = \sigma \circ L$ and $T$ leaves invariant a segment $\overline{xy}$ positive length. Suppose first that $L$ leaves invariant the same segment as well. By Lemma 3.2, a pseudo-orbit in the interior of the hypercube $I^n$ cannot be traced by the orbit of a point in the hypercube. If $L$ moves the segment $\overline{xy}$ invariant under $T$, we can assume without loss of generality it lies entirely on a hyperplane face $F$ of $I^n$ and the action of $\sigma$ on $L(\overline{xy})$ is just a projection over $F$. But in that case, the action of $T$ on the segment is a (composition of two) linear map(s) and the same argument applies. $\square$

We point out that, in particular, $T$ may not be observable even if $W$ is hyperbolic.

The condition in Theorem 3.3 is, in fact, sufficient. The proof is more involved and is given in detail in (Garzon & Botelho, 1994). WIth Theorem 3.3 one can then determine relatively simple necessary and sufficient conditions for observability (in terms of the eigenvalues and determinants of a finite number of linear maps). They establish Theorem 2.1 for saturated-linear activation functions.

## 4   BOOLEAN NETWORKS

This section contains precise definitions of discrete (boolean) neural networks and a sketch of the proof that they are observable in general.

Discrete neural networks have a finite number of activations and their state sets are endowed with an addition and multiplication. The activation function $\delta_i$ (typically

a threshold function) can be given by an arbitrary boolean table, which may vary from cell to cell. They can, moreover, have an infinite number of cells (the only case of interest here, since finite booolean networks are trivially observable). However, since the activation set if is finite, it only makes sense to consider *locally finite* networks, for which every cell $i$ only receives input from finitely many others.

A total state is now usually called a *configuration*. A configuration is best thought of as a bi-infinite sequence $x := x_1 x_2 x_3 \cdots$ consisting of the activations of all cells listed in some fixed order. The space of all configurations is a compact metric space if endowed with any of a number of equivalent metrics, such as $|x, y| := \frac{1}{2^m}$, where $m = \inf\{i : x_i \neq y_i\}$. In this metric, a small perturbation of a configuration is obtained by changing the values of $x$ at pixels far away from $x_1$.

The simplest question about observability in a general space concerns the shadowing of the identity function. Observability of the identity happens to be a property characteristic of configuration spaces. Recall that a totally disconnected topological space is one in which the connected component of every element is itself.

**Theorem 4.1** *The identity map* id *of a compact metric space $X$ is observable iff $X$ is totally disconnected.*

The first step in the proof of Theorem 4.3 below is to characterize observability of linear boolean networks (i.e. those obeying the superposition principle).

**Theorem 4.2** *Every linear continuous map has the shadowing property.*

For the other step we use a global decomposition $T = F \circ L$ of the global dynamics of a discrete network as a continuous transformation of configuration space due to (Garzon & Franklin, 1990). The reader is referred to (Garzon & Botelho, 1992) for a detailed proof of all the results in this section.

**Theorem 4.3** *Every discrete (boolean) neural network is observable.*

## 5    CONCLUSION AND OPEN PROBLEMS

It has been shown that the particular combination of a linear map with an activation function is usually globally observable, despite the fact that neither of them is observable and the fact that, ordinarily, composition destroys observability. Intuitively, this means that observing the input/output behavior of a neural network will eventually give away the true nature of the network's behavior, even if the network perturbs its behavior slighlty at each step of its evolution. In simple terms, such a network cannot fool all the people all of the time.

The results are valid for virtually every type of autonomous first-order network that evolves in discrete-time, whether the activations are boolean or continuous. Several results follow from this characterization. For example, in all likelihood there exist *observable* universal neural nets, despite the consequent undecidability of their computational behavior. Also, neural nets are thus a very natural set of primitives for approximation and implementation of more general *dynamical systems*. These and other consequences will be explored elsewhere (Botelho & Garzon, 1994).

Natural questions arise from these results. First, whether observability is a general property of most analog networks evolving in continuous time as well. Second, what other type of combinations of nonobservable systems of more general types creates observability, i.e. to what extent neural networks are peculiar in this regard. For example, are higher-order neural networks observable? Those with sigma-pi units? Finally, there is the broader question of robustness of neural network implementations, which bring about inevitable errors in input and/or weights. The results in this paper give a deeper explanation for the touted robustness and fault-tolerance of neural network solutions. But, further, they also seem to indicate that it may be possible to require that neural net solutions have observable behavior as well, without a tradeoff in the quality of the solution. An exact formulation of this question is worthy of further research.

## Acknowledgements

The work of the first author was partially done while on support from NSF grant CCR-9010985 and CNRS-France.

## References

F. Albertini and E.D. Sontag. (1993) Identifiability of discrete-time neural networks. In *Proc. European Control Conference*, 460-465. Groningen, The Netherlands: Springer-Verlag.

F. Albertini and E.D. Sontag. (1993) For neural networks, function determines form. *Neural Networks* 6(7): 975-990.

F. Botelho and M. Garzon. (1992) Boolean Neural Nets are Observable, Memphis State University: Technical Report 92-18.

F. Botelho and M. Garzon. (1994) Generalized Shadowing Properties. *J. Random and Computational Dynamics*, in print.

R. Bowen. (1978) On Axiom A diffeomorphisms. In *CBMS Regional Conference Series in Math.* **35**. Providence, Rhode Island: American Math. Society.

A.M. Chen, H. Lu, and R. Hecht-Nielsen, (1993) On the Geometry of Feedforward Neural Network Error Surfaces. *Neural Computation* 5(6): 910-927.

E. Coven, I. Kan, and J. Yorke. (1988) Pseudo-orbit shadowing in the family of tent maps. *Trans. AMS* **308**: 227-241.

M. Garzon and S. P. Franklin. (1990) Global dynamics in neural networks II. *Complex Systems* 4(5): 509-518.

M. Garzon and F. Botelho. (1994) Observability of Discrete-time Analog Networks, preprint.

E.D. Sontag. (1990) *Mathematical Control Theory: Deterministic Finite-Dimensional Dynamical Systems*. New York: Springer-Verlag.

H. Sussmann. (1992) Uniqueness of the Weights for Minimal Feedforward Nets with a Given Input-Output Map. *Neural Networks* 5(4): 589-593.
